# Local Algorithms for Approximate Inference in Minor-Excluded Graphs

**Kyomin Jung**
Dept. of Mathematics, MIT
kmjung@mit.edu

**Devavrat Shah**
Dept. of EECS, MIT
devavrat@mit.edu

## Abstract

We present a new local approximation algorithm for computing MAP and log-partition function for arbitrary exponential family distribution represented by a finite-valued pair-wise Markov random field (MRF), say $G$. Our algorithm is based on decomposing $G$ into *appropriately* chosen small components; computing estimates locally in each of these components and then producing a *good* global solution. We prove that the algorithm can provide approximate solution within *arbitrary accuracy* when $G$ excludes some finite sized graph as its minor and $G$ has bounded degree: all Planar graphs with bounded degree are examples of such graphs. The running time of the algorithm is $\Theta(n)$ ($n$ is the number of nodes in $G$), with constant dependent on accuracy, degree of graph and size of the graph that is excluded as a minor (constant for Planar graphs).

Our algorithm for minor-excluded graphs uses the decomposition scheme of Klein, Plotkin and Rao (1993). In general, our algorithm works with any decomposition scheme and provides quantifiable approximation guarantee that depends on the decomposition scheme.

## 1 Introduction

Markov Random Field (MRF) based exponential family of distribution allows for representing distributions in an intuitive parametric form. Therefore, it has been successful for modeling in many applications Specifically, consider an exponential family on $n$ random variables $\mathbf{X} = (X_1, \ldots, X_n)$ represented by a pair-wise (undirected) MRF with graph structure $G = (V, E)$, where vertices $V = \{1, \ldots, n\}$ and edge set $E \subset V \times V$. Each $X_i$ takes value in a finite set $\Sigma$ (e.g. $\Sigma = \{0, 1\}$). The joint distribution of $\mathbf{X} = (X_i)$: for $\mathbf{x} = (x_i) \in \Sigma^n$,

$$\Pr[\mathbf{X} = \mathbf{x}] \quad \propto \quad \exp\left(\sum_{i \in V} \phi_i(x_i) + \sum_{(i,j) \in E} \psi_{ij}(x_i, x_j)\right). \tag{1}$$

Here, functions $\phi_i : \Sigma \to \mathbb{R}^+ \stackrel{\triangle}{=} \{x \in \mathbb{R} : x \geq 0\}$, and $\psi_{ij} : \Sigma^2 \to \mathbb{R}^+$ are assumed to be arbitrary non-negative (real-valued) functions.[1] The two most important computational questions of interest are: (i) finding maximum a-posteriori (MAP) assignment $\mathbf{x}^*$, where $\mathbf{x}^* = \arg\max_{\mathbf{x} \in \Sigma^n} \Pr[\mathbf{X} = \mathbf{x}]$; and (ii) marginal distributions of variables, i.e. $\Pr[X_i = x]$; for $x \in \Sigma, 1 \leq i \leq n$. MAP is equivalent to a *minimal energy assignment* (or ground state) where energy, $\mathcal{E}(\mathbf{x})$, of state $\mathbf{x} \in \Sigma^n$ is defined as $\mathcal{E}(\mathbf{x}) = -\mathcal{H}(\mathbf{x}) + \mathsf{Constant}$, where $\mathcal{H}(\mathbf{x}) = \sum_{i \in V} \phi_i(x_i) + \sum_{(i,j) \in E} \psi_{ij}(x_i, x_j)$. Similarly, computing marginal is equivalent to computing log-partition function, defined as $\log Z = \log\left(\sum_{\mathbf{x} \in \Sigma^n} \exp\left(\sum_{i \in V} \phi_i(x_i) + \sum_{(i,j) \in E} \psi_{ij}(x_i, x_j)\right)\right)$. In this paper, we will find $\varepsilon$-approximation solutions of MAP and log-partition function: that is, $\hat{\mathbf{x}}$ and $\log \hat{Z}$ such that: $(1 - \varepsilon)\mathcal{H}(\mathbf{x}^*) \leq \mathcal{H}(\hat{\mathbf{x}}) \leq \mathcal{H}(\mathbf{x}^*)$, $(1 - \varepsilon)\log Z \leq \log \hat{Z} \leq (1 + \varepsilon)\log Z$.

**Previous Work.** The question of finding MAP (or ground state) comes up in many important application areas such as coding theory, discrete optimization, image denoising.Similarly, log-partition function is used in counting combinatorial objects loss-probability computation in computer networks, etc. Both problems are NP-hard for exact and even (constant) approximate computation for arbitrary graph $G$. However, applications require solving this problem using very simple algorithms. A plausible approach is as follows. First, identify wide class of graphs that have simple algorithms for computing MAP and log-partition function. Then, try to build system (e.g. codes) so that such good graph structure emerges and use the simple algorithm or else use the algorithm as a heuristic.

Such an approach has resulted in many interesting recent results starting the Belief Propagation (BP) algorithm designed for Tree graph [1].Since there a vast literature on this topic, we will recall only few results. Two important algorithms are the generalized belief propagation (BP) [2] and the tree-reweighted algorithm (TRW) [3,4].Key properties of interest for these iterative procedures are the correctness of fixed points and convergence. Many results characterizing properties of the fixed points are known starting from [2]. Various sufficient conditions for their convergence are known starting [5]. However, simultaneous convergence and correctness of such algorithms are established for only specific problems, e.g. [6].

Finally, we discuss two relevant results. The first result is about properties of TRW. The TRW algorithm provides provable upper bound on log-partition function for arbitrary graph [3]However, to the best of authors' knowledge the error is not quantified. The TRW for MAP estimation has a strong connection to specific Linear Programming (LP) relaxation of the problem [4]. This was made precise in a sequence of work by Kolmogorov [7], Kolmogorov and Wainwright [8] for binary MRF. It is worth noting that LP relaxation can be poor even for simple problems.

The second is an approximation algorithm proposed by Globerson and Jaakkola [9] to compute log-partition function using Planar graph decomposition (PDC). PDC uses techniques of [3] in conjunction with known result about exact computation of partition function for binary MRF when $G$ is Planar and the exponential family has specific form. Their algorithm provides provable upper bound for arbitrary graph. However, they do not quantify the error incurred. Further, their algorithm is limited to binary MRF.

**Contribution.** We propose a novel local algorithm for approximate computation of MAP and log-partition function. For any $\varepsilon > 0$, our algorithm can produce an $\varepsilon$-approximate solution for MAP and log-partition function for *arbitrary* MRF $G$ as long as $G$ excludes a finite graph as a minor (precise definition later). For example, Planar graph excludes $K_{3,3}, K_5$ as a minor. The running time of the algorithm is $\Theta(n)$, with constant dependent on $\varepsilon$, the maximum vertex degree of $G$ and the size of the graph that is excluded as minor. Specifically, for a Planar graph with bounded degree, it takes $\leq C(\varepsilon)n$ time to find $\varepsilon$-approximate solution with $\log \log C(\varepsilon) = O(1/\varepsilon)$. In general, our algorithm works for any $G$ and we can quantify bound on the error incurred by our algorithm. It is worth noting that our algorithm provides a provable lower bound on log-partition function as well unlike many of previous works.

The precise results for minor-excluded graphs are stated in Theorems 1 and 2. The result concerning general graphs are stated in the form of Lemmas 2-3-4 for log-partition and Lemmas 5-6-7 for MAP.

**Techniques.** Our algorithm is based on the following idea: First, decompose $G$ into small-size connected components say $G_1, \ldots, G_k$ by removing few edges of $G$. Second, compute estimates (either MAP or log-partition) in each of $G_i$ separately. Third, combine these estimates to produce a global estimate while *taking care* of the effect induced by removed edges. We show that the error in the estimate depends only on the edges removed. This error bound characterization is applicable for arbitrary graph.

Klein, Plotkin and Rao [10]introduced a clever and simple decomposition method for minor-excluded graphs to study the gap between max-flow and min-cut for multicommodity flows. We use their method to obtain a good edge-set for decomposing minor-excluded $G$ so that the error induced in our estimate is small (can be made as small as required).

In general, as long as $G$ allows for such good edge-set for decomposing $G$ into small components, our algorithm will provide a good estimate. To compute estimates in individual components, we use dynamic programming. Since each component is small, it is not computationally burdensome.

However, one may obtain further simpler heuristics by replacing dynamic programming by other method such as BP or TRW for computation in the components.

## 2 Preliminaries

Here we present useful definitions and previous results about decomposition of minor-excluded graphs from [10,11].

**Definition 1 (Minor Exclusion)** *A graph $H$ is called minor of $G$ if we can transform $G$ into $H$ through an arbitrary sequence of the following two operations: (a) removal of an edge; (b) merge two connected vertices $u, v$: that is, remove edge $(u, v)$ as well as vertices $u$ and $v$; add a new vertex and make all edges incident on this new vertex that were incident on $u$ or $v$. Now, if $H$ is not a minor of $G$ then we say that $G$ excludes $H$ as a minor.*

The explanation of the following statement may help understand the definition: *any graph $H$ with $r$ nodes is a minor of $K_r$*, where $K_r$ is a complete graph of $r$ nodes. This is true because one may obtain $H$ by removing edges from $K_r$ that are absent in $H$. More generally, if $G$ is a subgraph of $G'$ and $G$ has $H$ as a minor, then $G'$ has $H$ as its minor. Let $K_{r,r}$ denote a complete bipartite graph with $r$ nodes in each partition. Then $K_r$ is a minor of $K_{r,r}$. An important implication of this is as follows: to prove property P for graph $G$ that excludes $H$, of size $r$, as a minor, it is sufficient to prove that any graph that excludes $K_{r,r}$ as a minor has property P. This fact was cleverly used by Klein et. al. [10] to obtain a good decomposition scheme described next. First, a definition.

**Definition 2 ($(\delta, \Delta)$-decomposition)** *Given graph $G = (V, E)$, a randomly chosen subset of edges $\mathcal{B} \subset E$ is called $(\delta, \Delta)$ decomposition of $G$ if the following holds: (a) For any edge $e \in E$, $\Pr(e \in \mathcal{B}) \leq \delta$. (b) Let $S_1, \ldots, S_K$ be connected components of graph $G' = (V, E \backslash \mathcal{B})$ obtained by removing edges of $\mathcal{B}$ from $G$. Then, for any such component $S_j, 1 \leq j \leq K$ and any $u, v \in S_j$ the shortest-path distance between $(u, v)$ in the original graph $G$ is at most $\Delta$ with probability $1$.*

The existence of $(\delta, \Delta)$-decomposition implies that it is possible to remove $\delta$ fraction of edges so that graph *decomposes* into connected components whose *diameter* is small. We describe a simple and explicit construction of such a decomposition for minor excluded class of graphs. This scheme was proposed by Klein, Plotkin, Rao [10] and Rao [11].

---

**DeC**$(G, r, \Delta)$

---

    (0) Input is graph $G = (V, E)$ and $r, \Delta \in \mathbb{N}$. Initially, $i = 0, G_0 = G, \mathcal{B} = \emptyset$.

    (1) For $i = 0, \ldots, r - 1$, do the following.

        (a) Let $S_1^i, \ldots, S_{k_i}^i$ be the connected components of $G_i$.

        (b) For each $S_j^i, 1 \leq j \leq k_i$, pick an arbitrary node $v_j \in S_j^i$.

            ◦ Create a breadth-first search tree $\mathcal{T}_j^i$ rooted at $v_j$ in $S_j^i$.

            ◦ Choose a number $L_j^i$ uniformly at random from $\{0, \ldots, \Delta - 1\}$.

            ◦ Let $\mathcal{B}_j^i$ be the set of edges at level $L_j^i, \Delta + L_j^i, 2\Delta + L_j^i, \ldots$ in $\mathcal{T}_j^i$.

            ◦ Update $\mathcal{B} = \mathcal{B} \cup_{j=1}^{k_i} \mathcal{B}_j^i$.

        (c) set $i = i + 1$.

    (3) Output $\mathcal{B}$ and graph $G' = (V, E \backslash \mathcal{B})$.

---

As stated above, the basic idea is to use the following step recursively (upto depth $r$ of recursion): in each connected component, say $S$, choose a node arbitrarily and create a breadth-first search tree, say $\mathcal{T}$. Choose a number, say $L$, uniformly at random from $\{0, \ldots, \Delta - 1\}$. Remove (add to $\mathcal{B}$) all edges that are at level $L + k\Delta, k \geq 0$ in $\mathcal{T}$. Clearly, the total running time of such an algorithm is $O(r(n + |E|))$ for a graph $G = (V, E)$ with $|V| = n$; with possible parallel implementation across different connected components.

The algorithm **DeC**$(G, r, \Delta)$ is designed to provide a good decomposition for class of graphs that exclude $K_{r,r}$ as a minor. Figure 1 explains the algorithm for a line-graph of $n = 9$ nodes, which excludes $K_{2,2}$ as a minor. The example is about a sample run of **DeC**$(G, 2, 3)$ (Figure 1 shows the first iteration of the algorithm).

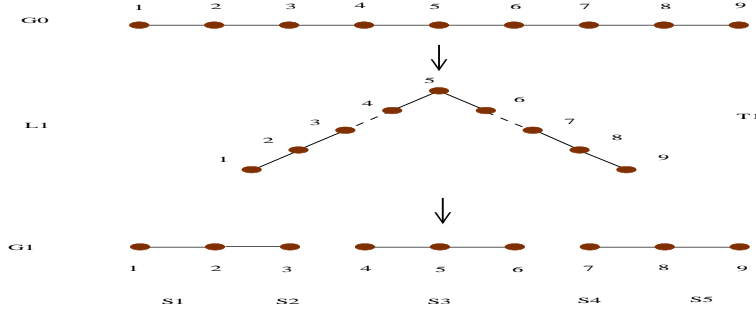

Figure 1: The first of two iterations in execution of **DeC**$(G, 2, 3)$ is shown.

**Lemma 1** *If $G$ excludes $K_{r,r}$ as a minor, then algorithm* **DeC**$(G, r, \Delta)$ *outputs $\mathcal{B}$ which is $(r/\Delta, O(\Delta))$-decomposition of $G$.*

It is known that Planar graph excludes $K_{3,3}$ as a minor. Hence, Lemma 1 implies the following.

**Corollary 1** *Given a planar graph $G$, the algorithm* **DeC**$(G, 3, \Delta)$ *produces $(3/\Delta, O(\Delta))$-decomposition for any $\Delta \geq 1$.*

## 3   Approximate $\log Z$

Here, we describe algorithm for approximate computation of $\log Z$ for any graph $G$. The algorithm uses a decomposition algorithm as a sub-routine. In what follows, we use term DECOMP for a generic decomposition algorithm. The key point is that our algorithm provides provable upper and lower bound on $\log Z$ for any graph; the approximation guarantee and computation time depends on the property of DECOMP. Specifically, for $K_{r,r}$ minor excluded $G$ (e.g. Planar graph with $r = 3$), we will use **DeC**$(G, r, \Delta)$ in place of DECOMP. Using Lemma 1, we show that our algorithm based on **DeC** provides approximation upto arbitrary multiplicative accuracy by tuning parameter $\Delta$.

LOG PARTITION$(G)$

---

(1) Use DECOMP$(G)$ to obtain $\mathcal{B} \subset E$ such that
   (a) $G' = (V, E \backslash \mathcal{B})$ is made of connected components $S_1, \ldots, S_K$.
(2) For each connected component $S_j, 1 \leq j \leq K$, do the following:
   (a) Compute partition function $Z_j$ restricted to $S_j$ by dynamic programming(or exhaustive computation).
(3) Let $\psi_{ij}^L = \min_{(x,x') \in \Sigma^2} \psi_{ij}(x, x')$, $\psi_{ij}^U = \max_{(x,x') \in \Sigma^2} \psi_{ij}(x, x')$. Then
$$\log \hat{Z}_{\text{LB}} = \sum_{j=1}^{K} \log Z_j + \sum_{(i,j) \in \mathcal{B}} \psi_{ij}^L; \quad \log \hat{Z}_{\text{UB}} = \sum_{j=1}^{K} \log Z_j + \sum_{(i,j) \in \mathcal{B}} \psi_{ij}^U.$$
(4) Output: lower bound $\log \hat{Z}_{\text{LB}}$ and upper bound $\log \hat{Z}_{\text{UB}}$.

---

In words, LOG PARTITION$(G)$ produces upper and lower bound on $\log Z$ of MRF $G$ as follows: decompose graph $G$ into (small) components $S_1, \ldots, S_K$ by removing (few) edges $\mathcal{B} \subset E$ using DECOMP$(G)$. Compute exact log-partition function in each of the components. To produce bounds $\log \hat{Z}_{\text{LB}}, \log \hat{Z}_{\text{UB}}$ take the summation of thus computed component-wise log-partition function along with minimal and maximal effect of edges from $\mathcal{B}$.

**Analysis of** LOG PARTITION **for General** $G$ **:**   Here, we analyze performance of LOG PARTITION for any $G$. In the next section, we will specialize our analysis for minor excluded $G$ when LOG PARTITION uses **DeC** as the DECOMP algorithm.

**Lemma 2** *Given an MRF $G$ described by (1), the* LOG PARTITION *produces $\log \hat{Z}_{LB}, \log \hat{Z}_{UB}$ such that*
$$\log \hat{Z}_{LB} \leq \log Z \leq \log \hat{Z}_{UB}, \quad \log \hat{Z}_{UB} - \log \hat{Z}_{LB} = \sum_{(i,j) \in \mathcal{B}} \left( \psi_{ij}^U - \psi_{ij}^L \right).$$

*It takes $O\left(|E|K\Sigma^{|S^*|}\right) + T_{\text{DECOMP}}$ time to produce this estimate, where $|S^*| = \max_{j=1}^{K} |S_j|$ with* DECOMP *producing decomposition of $G$ into $S_1, \ldots, S_K$ in time $T_{\text{DECOMP}}$.*

**Lemma 3** *If $G$ has maximum vertex degree $D$ then, $\log Z \geq \frac{1}{D+1}\left[\sum_{(i,j)\in E} \psi_{ij}^U - \psi_{ij}^L\right]$.*

**Lemma 4** *If $G$ has maximum vertex degree $D$ and the* DECOMP($G$) *produces $\mathcal{B}$ that is $(\delta, \Delta)$-decomposition, then*

$$\mathbb{E}\left[\log \hat{Z}_{UB} - \log \hat{Z}_{LB}\right] \leq \delta(D+1)\log Z,$$

*w.r.t. the randomness in $\mathcal{B}$, and* LOG PARTITION *takes time $O(nD|\Sigma|^{D^\Delta}) + T_{\text{DECOMP}}$.*

**Analysis of** LOG PARTITION **for Minor-excluded $G$ :** Here, we specialize analysis of LOG PARTITIONfor minor exclude graph $G$. For $G$ that exclude minor $K_{r,r}$, we use algorithm **DeC**($G, r, \Delta$). Now, we state the main result for log-partition function computation.

**Theorem 1** *Let $G$ exclude $K_{r,r}$ as minor and have $D$ as maximum vertex degree. Given $\varepsilon > 0$, use* LOG PARTITION *algorithm with* **DeC***($G, r, \Delta$) where $\Delta = \lceil \frac{r(D+1)}{\varepsilon} \rceil$. Then,*

$$\log \hat{Z}_{LB} \leq \log Z \leq \log \hat{Z}_{UB}; \qquad \mathbb{E}\left[\log \hat{Z}_{UB} - \log \hat{Z}_{LB}\right] \leq \varepsilon \log Z.$$

*Further, algorithm takes $(nC(D, |\Sigma|, \varepsilon))$, where constant $C(D, |\Sigma|, \varepsilon) = D|\Sigma|^{D^{O(rD/\varepsilon)}}$.*

We obtain the following immediate implication of Theorem 1.

**Corollary 2** *For any $\varepsilon > 0$, the* LOG PARTITION *algorithm with* **DeC** *algorithm for constant degree Planar graph $G$ based MRF, produces $\log \hat{Z}_{LB}, \log \hat{Z}_{UB}$ so that*

$$(1-\varepsilon)\log Z \leq \log \hat{Z}_{LB} \leq \log Z \leq \log \hat{Z}_{UB} \leq (1+\varepsilon)\log Z,$$

*in time $O(nC(\varepsilon))$ where $\log \log C(\varepsilon) = O(1/\varepsilon)$.*

## 4 Approximate MAP

Now, we describe algorithm to compute MAP approximately. It is very similar to the LOG PARTITION algorithm: given $G$, decompose it into (small) components $S_1, \ldots, S_K$ by removing (few) edges $\mathcal{B} \subset E$. Then, compute an approximate MAP assignment by computing exact MAP restricted to the components. As in LOG PARTITION, the computation time and performance of the algorithm depends on property of decomposition scheme. We describe algorithm for any graph $G$; which will be specialized for $K_{r,r}$ minor excluded $G$ using **DeC**($G, r, \Delta$).

MODE($G$)

---

(1) Use DECOMP($G$) to obtain $\mathcal{B} \subset E$ such that
   (a) $G' = (V, E \backslash \mathcal{B})$ is made of connected components $S_1, \ldots, S_K$.
(2) For each connected component $S_j, 1 \leq j \leq K$, do the following:
   (a) Through dynamic programming (or exhaustive computation) find exact MAP $\mathbf{x}^{*,j}$ for component $S_j$, where $\mathbf{x}^{*,j} = (x_i^{*,j})_{i \in S_j}$.
(3) Produce output $\widehat{\mathbf{x}^*}$, which is obtained by assigning values to nodes using $\mathbf{x}^{*,j}, 1 \leq j \leq K$.

---

**Analysis of** MODE **for General $G$ :** Here, we analyze performance of MODE for any $G$. Later, we will specialize our analysis for minor excluded $G$ when it uses **DeC** as the DECOMP algorithm.

**Lemma 5** *Given an MRF $G$ described by (1), the* MODE *algorithm produces outputs $\widehat{\mathbf{x}^*}$ such that $\mathcal{H}(\mathbf{x}^*) - \sum_{(i,j)\in\mathcal{B}} \left(\psi_{ij}^U - \psi_{ij}^L\right) \leq \mathcal{H}(\widehat{\mathbf{x}^*}) \leq \mathcal{H}(\mathbf{x}^*)$. It takes $O\left(|E|K\Sigma^{|S^*|}\right) + T_{\text{DECOMP}}$ time to produce this estimate, where $|S^*| = \max_{j=1}^{K} |S_j|$ with* DECOMP *producing decomposition of $G$ into $S_1, \ldots, S_K$ in time $T_{\text{DECOMP}}$.*

**Lemma 6** *If $G$ has maximum vertex degree $D$, then*

$$\mathcal{H}(\mathbf{x}^*) \geq \frac{1}{D+1}\left[\sum_{(i,j)\in E} \psi_{ij}^U\right] \geq \frac{1}{D+1}\left[\sum_{(i,j)\in E} \psi_{ij}^U - \psi_{ij}^L\right].$$

**Lemma 7** *If $G$ has maximum vertex degree $D$ and the* DECOMP*(G) produces $\mathcal{B}$ that is $(\delta, \Delta)$-decomposition, then*

$$\mathbb{E}\left[\mathcal{H}(\mathbf{x}^*) - \mathcal{H}(\widehat{\mathbf{x}^*})\right] \le \delta(D+1)\mathcal{H}(\mathbf{x}^*),$$

*where expectation is w.r.t. the randomness in $\mathcal{B}$. Further,* MODE *takes time $O(nD|\Sigma|^{D^\Delta}) + T_{\text{DECOMP}}$.*

**Analysis of** MODE **for Minor-excluded** $G$ **:** Here, we specialize analysis of MODE for minor exclude graph $G$. For $G$ that exclude minor $K_{r,r}$, we use algorithm **DeC**$(G, r, \Delta)$. Now, we state the main result for MAP computation.

**Theorem 2** *Let $G$ exclude $K_{r,r}$ as minor and have $D$ as the maximum vertex degree. Given $\varepsilon > 0$, use* MODE *algorithm with* **DeC**$(G, r, \Delta)$ *where $\Delta = \lceil \frac{r(D+1)}{\varepsilon} \rceil$. Then,*

$$(1 - \varepsilon)\mathcal{H}(\mathbf{x}^*) \le \mathcal{H}(\widehat{\mathbf{x}^*}) \le \mathcal{H}(\mathbf{x}^*).$$

*Further, algorithm takes $n \cdot C(D, |\Sigma|, \varepsilon)$ time, where constant $C(D, |\Sigma|, \varepsilon) = D|\Sigma|^{D^{O(rD/\varepsilon)}}$.*

We obtain the following immediate implication of Theorem 2.

**Corollary 3** *For any $\varepsilon > 0$, the* MODE *algorithm with* **DeC** *algorithm for constant degree Planar graph $G$ based MRF, produces estimate $\widehat{\mathbf{x}^*}$ so that*

$$(1 - \varepsilon)\mathcal{H}(\mathbf{x}^*) \le \mathcal{H}(\widehat{\mathbf{x}^*}) \le \mathcal{H}(\mathbf{x}^*),$$

*in time $O(nC(\varepsilon))$ where $\log \log C(\varepsilon) = O(1/\varepsilon)$.*

## 5 Experiments

Our algorithm provides provably good approximation for any MRF with minor excluded graph structure, with planar graph as a special case. In this section, we present experimental evaluation of our algorithm for popular synthetic model.

**Setup 1.**[2] Consider binary (i.e. $\Sigma = \{0, 1\}$) MRF on an $n \times n$ lattice $G = (V, E)$:

$$\Pr(\mathbf{x}) \propto \exp\left(\sum_{i \in V} \theta_i x_i + \sum_{(i,j) \in E} \theta_{ij} x_i x_j\right), \text{ for } \mathbf{x} \in \{0, 1\}^{n^2}.$$

Figure 2 shows a lattice or grid graph with $n = 4$ (on the left side). There are two scenarios for choosing parameters (with notation $\mathcal{U}[a, b]$ being uniform distribution over interval $[a, b]$):

(1) *Varying interaction.* $\theta_i$ is chosen independently from distribution $\mathcal{U}[-0.05, 0.05]$ and $\theta_{ij}$ chosen independent from $\mathcal{U}[-\alpha, \alpha]$ with $\alpha \in \{0.2, 0.4, \ldots, 2\}$.

(2) *Varying field.* $\theta_{ij}$ is chosen independently from distribution $\mathcal{U}[-0.5, 0.5]$ and $\theta_i$ chosen independently from $\mathcal{U}[-\alpha, \alpha]$ with $\alpha \in \{0.2, 0.4, \ldots, 2\}$.

The grid graph is planar. Hence, we run our algorithms LOG PARTITION and MODE, with decomposition scheme **DeC**$(G, 3, \Delta)$, $\Delta \in \{3, 4, 5\}$. We consider two measures to evaluate performance: error in $\log Z$, defined as $\frac{1}{n^2}|\log Z^{\text{alg}} - \log Z|$; and error in $\mathcal{H}(\mathbf{x}^*)$, defined as $\frac{1}{n^2}|\mathcal{H}(\mathbf{x}^{\text{alg}} - \mathcal{H}(\mathbf{x}^*)|$.

We compare our algorithm for error in $\log Z$ with the two recently very successful algorithms – Tree re-weighted algorithm (TRW) and planar decomposition algorithm (PDC). The comparison is plotted in Figure 3 where $n = 7$ and results are averages over 40 trials. The Figure 3(A) plots error with respect to varying interaction while Figure 3(B) plots error with respect to varying field strength. Our algorithm, essentially outperforms TRW for these values of $\Delta$ and perform very competitively with respect to PDC.

The key feature of our algorithm is scalability. Specifically, running time of our algorithm with a given parameter value $\Delta$ scales linearly in $n$, while keeping the relative error bound exactly the same. To explain this important feature, we plot the theoretically evaluated bound on error in $\log Z$

in Figure 4 with tags (A), (B) and (C). Note that error bound plot is the same for $n = 100$ (A) and $n = 1000$ (B). Clearly, actual error is likely to be smaller than these theoretically plotted bounds. We note that these bounds only depend on the interaction strengths and *not* on the values of fields strengths (C).

Results similar to of Log Partition are expected from Mode. We plot the theoretically evaluated bounds on the error in MAP in Figure 4 with tags (A), (B) and (C). Again, the bound on MAP relative error for given $\Delta$ parameter remains the same for all values of $n$ as shown in (A) for $n = 100$ and (B) for $n = 1000$. There is no change in error bound with respect to the field strength (C).

**Setup 2.** Everything is exactly the same as the above setup with the only difference that grid graph is replaced by *cris-cross* graph which is obtained by adding extra four neighboring edges per node (exception of boundary nodes). Figure 2 shows cris-cross graph with $n = 4$ (on the right side). We again run the same algorithm as above setup on this graph. For cris-cross graph, we obtained its graph decomposition from the decomposition of its grid sub-graph. graph Though the cris-cross graph is not planar, due to the structure of the cris-cross graph it can be shown (proved) that the running time of our algorithm will remain the same (in order) and error bound will become only 3 times weaker than that for the grid graph ! We compute these theoretical error bounds for $\log Z$ and MAP which is plotted in Figure 5. This figure is similar to the Figure 4 for grid graph. This clearly exhibits the generality of our algorithm even beyond minor excluded graphs.

## Footnotes

[1]Here, we assume the positivity of $\phi_i$'s and $\psi_{ij}$'s for simplicity of analysis.

[2]Though this setup has $\phi_i, \psi_{ij}$ taking negative values, they are equivalent to the setup considered in the paper as the function values are lower bounded and hence *affine* shift will make them non-negative without changing the distribution.

### References

[1] J. Pearl, "Probabilistic Reasoning in Intelligent Systems: Networks of Plausible Inference," San Francisco, CA: Morgan Kaufmann, 1988.

[2] J. Yedidia, W. Freeman and Y. Weiss, "Generalized Belief Propagation," *Mitsubishi Elect. Res. Lab.*, TR-2000-26, 2000.

[3] M. J. Wainwright, T. Jaakkola and A. S. Willsky, "Tree-based reparameterization framework for analysis of sum-product and related algorithms," *IEEE Trans. on Info. Theory*, 2003.

[4] M. J. Wainwright, T. S. Jaakkola and A. S. Willsky, "MAP estimation via agreement on (hyper)trees: Message-passing and linear-programming approaches," *IEEE Trans. on Info. Theory*, 51(11), 2005.

[5] S. C. Tatikonda and M. I. Jordan, "Loopy Belief Propagation and Gibbs Measure," *Uncertainty in Artificial Intelligence*, 2002.

[6] M. Bayati, D. Shah and M. Sharma, "Maximum Weight Matching via Max-Product Belief Propagation," *IEEE ISIT*, 2005.

[7] V. Kolmogorov, "Convergent Tree-reweighted Message Passing for Energy Minimization," *IEEE Transactions on Pattern Analysis and Machine Intelligence*, 2006.

[8] V. Kolmogorov and M. Wainwright, "On optimality of tree-reweighted max-product message-passing," *Uncertainty in Artificial Intelligence*, 2005.

[9] A. Globerson and T. Jaakkola, "Bound on Partition function through Planar Graph Decomposition," *NIPS*, 2006.

[10] P. Klein, S. Plotkin and S. Rao, "Excluded minors, network decomposition, and multicommodity flow," *ACM STOC*, 1993.

[11] S. Rao, "Small distortion and volume preserving embeddings for Planar and Euclidian metrics," *ACM SCG*, 1999.

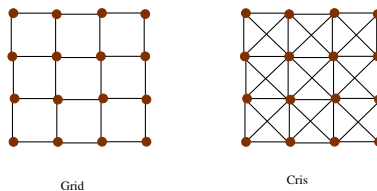

Figure 2: Example of grid graph (left) and cris-cross graph (right) with $n = 4$.

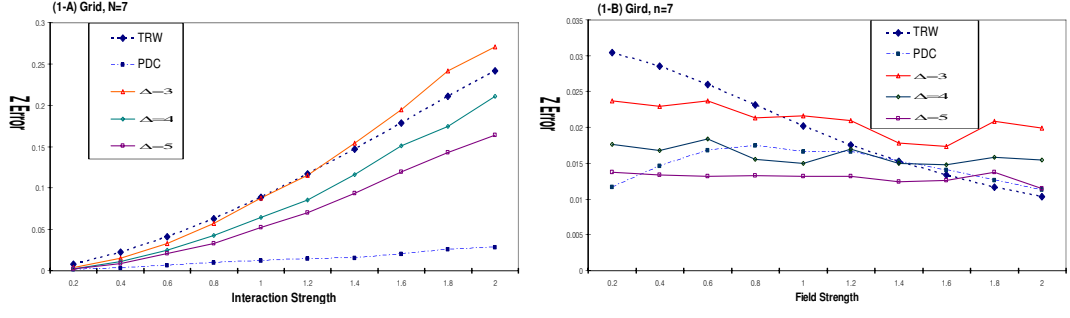

**Figure 3:** Comparison of TRW, PDC and our algorithm for grid graph with $n = 7$ with respect to error in $\log Z$. Our algorithm outperforms TRW and is competitive with respect to PDC.

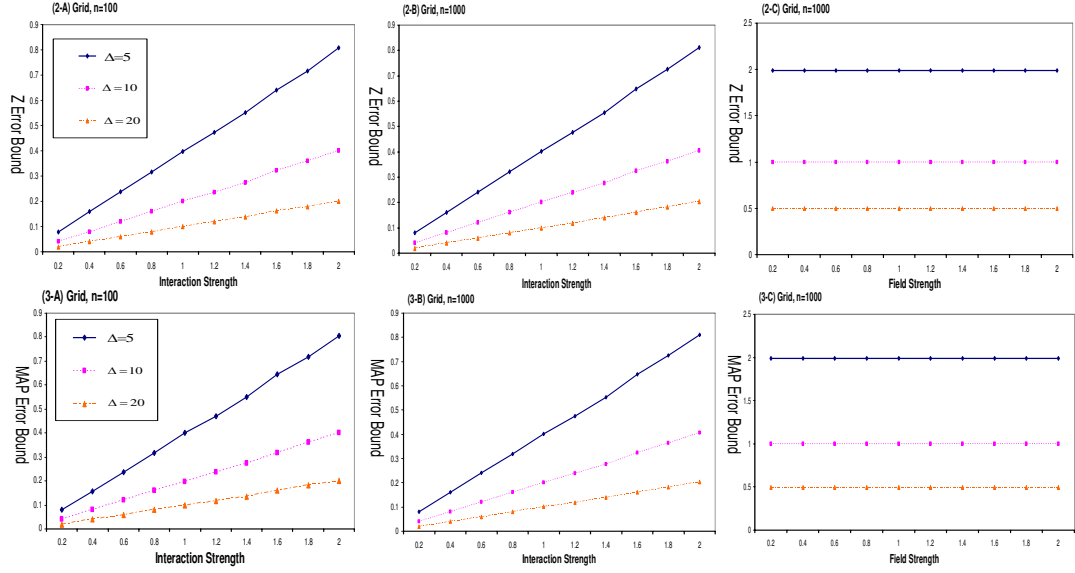

**Figure 4:** The theoretically computable error bounds for $\log Z$ and MAP under our algorithm for grid with $n = 100$ and $n = 1000$ under varying interaction and varying field model. This clearly shows scalability of our algorithm.

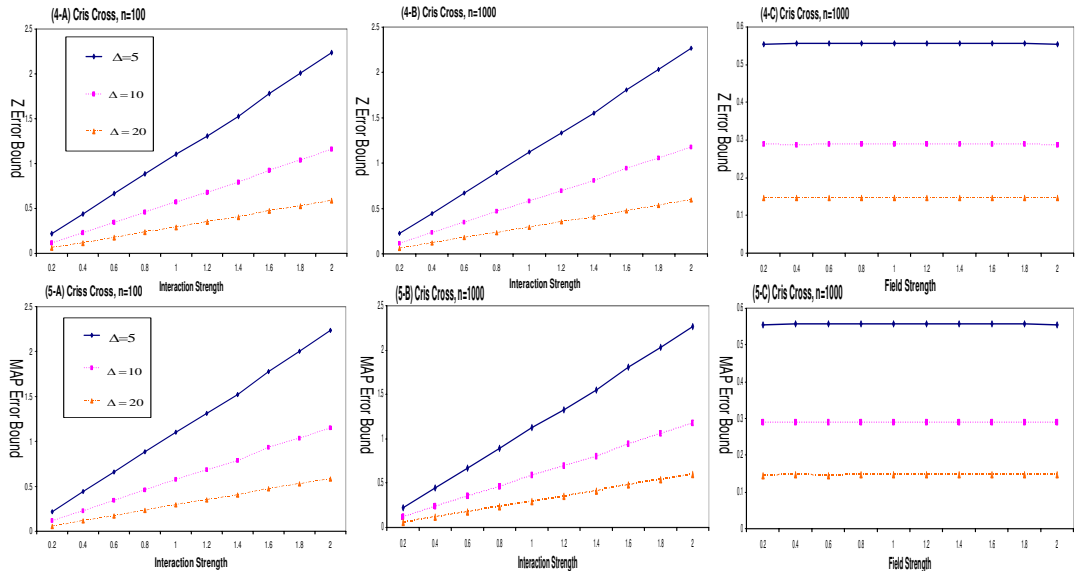

**Figure 5:** The theoretically computable error bounds for $\log Z$ and MAP under our algorithm for cris-cross with $n = 100$ and $n = 1000$ under varying interaction and varying field model. This clearly shows scalability of our algorithm and robustness to graph structure.

